# Efficient inference in matrix-variate Gaussian models with iid observation noise

**Oliver Stegle**[1]
Max Planck Institutes
Tübingen, Germany
stegle@tuebingen.mpg.de

**Christoph Lippert**[1]
Max Planck Institutes
Tübingen, Germany
clippert@tuebingen.mpg.de

**Joris Mooij**
Institute for Computing and Information Sciences
Radboud University
Nijmegen, The Netherlands
j.mooij@cs.ru.nl

**Neil Lawrence**
Department of Computer Science
University of Sheffield
Sheffield, UK
N.Lawrence@sheffield.ac.uk

**Karsten Borgwardt**
Max Planck Institutes & Eberhard Karls Universität
Tübingen, Germany
karsten.borgwardt@tuebingen.mpg.de

## Abstract

Inference in matrix-variate Gaussian models has major applications for multi-output prediction and joint learning of row and column covariances from matrix-variate data. Here, we discuss an approach for efficient inference in such models that explicitly account for iid observation noise. Computational tractability can be retained by exploiting the Kronecker product between row and column covariance matrices. Using this framework, we show how to generalize the Graphical Lasso in order to learn a sparse inverse covariance between features while accounting for a low-rank confounding covariance between samples. We show practical utility on applications to biology, where we model covariances with more than 100,000 dimensions. We find greater accuracy in recovering biological network structures and are able to better reconstruct the confounders.

## 1 Introduction

Matrix-variate normal (MVN) models have important applications in various fields. These models have been used as regularizer for multi-output prediction, jointly modeling the similarity between tasks and samples [1]. In related work in Gaussian processes (GPs), generalizations of MVN distributions have been used for inference of vector-valued functions [2, 3]. These models with Kronecker factored covariance have applications in geostatistics [4], statistical testing on matrix-variate data [5] and statistical genetics [6].

In prior work, different covariance functions for rows and columns have been combined in a flexible manner. For example, Dutilleul and Zhang et al. [7, 1] have performed estimation of free-form covariances with different norm penalties. In other applications for prediction [2] and dimension reduction [8], combinations of free-form covariances with squared exponential covariances have been used.

In the absence of iid observation noise, an efficient inference scheme also known as the "flip-flop algorithm" can be derived. In this iterative approach, estimation of the respective covariances is decoupled by rotating the data with respect to one of the covariances to optimize parameters of the other [7, 1]. While this simplifying assumption of noise-free matrix-variate data has been used with some success, there are clear motivations for including iid noise in the model. For example, Bonilla et al. [2] have shown that in multi-task regression a noise free GP with Kronecker structure leads to a cancelation of information sharing between the various prediction tasks. This effect, also known from the geostatistics literature [4], eliminates any benefit from multivariate prediction compared to naïve approaches. Alternatively, when including observation noise in the model, computational tractability has been limited to smaller datasets. The covariance matrix no longer directly factorizes into a Kronecker product, thus rendering simple approaches such as the "flip-flop algorithm" inappropriate.

Here, we address these shortcomings and propose a general framework for efficient inference in matrix-variate normal models that include iid observation noise. Although in this model the covariance matrix no longer factorizes into a Kronecker product, we show how efficient parameter inference can still be done. To this end, we provide derivations of both the log-likelihood and gradients with respect to hyperparameters that can be computed in the same asymptotic runtime as iterations of the "flip-flop algorithm" on a noise-free model. This allows for parameter learning of covariance matrices of size $10^5 \times 10^5$, or even bigger, which would not be possible if done naïvely.

First, we show how for any combination of covariances, evaluation of model likelihood and gradients with respect to individual covariance parameters is tractable. Second, we apply this framework to structure learning in Gaussian graphical models, while accounting for a confounding non-iid sample structure. This generalization of the *Graphical Lasso* [9, 10] (GLASSO) allows to jointly learn and account for a sparse inverse covariance matrix between features and a structured (non-diagonal) sample covariance. The low rank component of the sample covariance is used to account for confounding effects, as is done in other models for genomics [11, 12].

We illustrate this generalization called "Kronecker GLASSO" on synthetic datasets and heterogeneous protein signaling and gene expression data, where the aim is to recover the hidden network structures. We show that our approach is able to recover the confounding structure, when it is known, and reveals sparse biological networks that are in better agreement with known components of the latent network structure.

## 2  Efficient inference in Kronecker Gaussian processes

Assume we are given a data matrix $\mathbf{Y} \in \mathbb{R}^{N \times D}$ with $N$ rows and $D$ columns, where $N$ is the number of samples with $D$ features each. As an example, think of $N$ as a number of micro-array experiments, where in each experiment the expression levels of the same $D$ genes are measured; here, $y_{rc}$ would be the expression level of gene $c$ in experiment $r$. Alternatively, $\mathbf{Y}$ could represent multi-variate targets in a multi-task prediction setting, with rows corresponding to tasks and columns to features. This setting occurs in geostatistics, where the entries $y_{rc}$ correspond to ecological measurements taken on a regular grid.

First we introduce some notation. For any $L \times M$ matrix $\mathbf{A}$, we define $\mathrm{vec}(\mathbf{A})$ to be the vector obtained by concatenating the columns of $\mathbf{A}$; further, let $\mathbf{A} \otimes \mathbf{B}$ denote the *Kronecker product* (or *tensor product*) between matrices $\mathbf{A}$ and $\mathbf{B}$:

$$\mathrm{vec}(\mathbf{A}) = \begin{pmatrix} a_{11} \\ a_{21} \\ \vdots \\ a_{LM} \end{pmatrix}; \qquad \mathbf{A} \otimes \mathbf{B} = \begin{pmatrix} a_{11}\mathbf{B} & a_{12}\mathbf{B} & \dots & a_{1M}\mathbf{B} \\ a_{21}\mathbf{B} & a_{22}\mathbf{B} & \dots & a_{2M}\mathbf{B} \\ \dots & \dots & \dots & \vdots \\ a_{L1}\mathbf{B} & a_{L2}\mathbf{B} & \dots & a_{LM}\mathbf{B} \end{pmatrix}.$$

For modeling $\mathbf{Y}$ as a matrix-variate normal distribution with iid observation noise, we first introduce $N \times D$ additional latent variables $\mathbf{Z}$, which can be thought of as the noise-free observations. The data $\mathbf{Y}$ is then given by $\mathbf{Z}$ plus iid Gaussian observation noise:

$$p(\mathbf{Y} \,|\, \mathbf{Z}, \sigma^2) = \mathcal{N}\left(\mathrm{vec}(\mathbf{Y}) \,\middle|\, \mathrm{vec}(\mathbf{Z}), \sigma^2 \mathbf{I}_{N \cdot D}\right). \tag{1}$$

If the covariance between rows and columns of the noise-free observations $\mathbf{Z}$ factorizes, we may assume a zero-mean matrix-variate normal model for $\mathbf{Z}$:

$$p(\mathbf{Z} \,|\, \mathbf{C}, \mathbf{R}) = \frac{\exp\{-\frac{1}{2}\mathrm{Tr}[\mathbf{C}^{-1}\mathbf{Z}^\mathrm{T}\mathbf{R}^{-1}\mathbf{Z}]\}}{(2\pi)^{ND/2}|\mathbf{R}|^{N/2}|\mathbf{C}|^{D/2}},$$

which can be equivalently formulated as a multivariate normal distribution:

$$= \mathcal{N}\left(\mathrm{vec}(\mathbf{Z}) \,|\, \mathbf{0}_{N\cdot D}, \mathbf{C}(\mathbf{\Theta_C}) \otimes \mathbf{R}(\mathbf{\Theta_R})\right). \tag{2}$$

Here, the matrix $\mathbf{C}$ is a $D \times D$ column covariance matrix and $\mathbf{R}$ is an $N \times N$ row covariance matrix that may depend on hyperparameters $\mathbf{\Theta_C}$ and $\mathbf{\Theta_R}$ respectively. Marginalizing over the noise-free observations $\mathbf{Z}$ results in the Kronecker Gaussian process model of the observed data $\mathbf{Y}$

$$p(\mathbf{Y} \,|\, \mathbf{C}, \mathbf{R}, \sigma^2) = \mathcal{N}\left(\mathrm{vec}(\mathbf{Y}) \,\big|\, \mathbf{0}_{N\cdot D}, \mathbf{C}(\mathbf{\Theta_C}) \otimes \mathbf{R}(\mathbf{\Theta_R}) + \sigma^2\mathbf{I}_{N\cdot D}\right). \tag{3}$$

For notational convenience we will drop the dependency on hyperparameters $\mathbf{\Theta_C}$, $\mathbf{\Theta_R}$ and $\sigma^2$. Note that for $\sigma^2 = 0$, the likelihood model in Equation (3) reduces to the matrix-variate normal distribution in Equation (2).

## 2.1 Efficient parameter estimation

For efficient optimization of the log likelihood, $\mathcal{L} = \ln p(\mathbf{Y} \,|\, \mathbf{C}, \mathbf{R}, \sigma^2)$, with respect to the hyperparameters, we exploit an identity that allows us to write a matrix product with a Kronecker product matrix in terms of ordinary matrix products:

$$(\mathbf{C} \otimes \mathbf{R})\mathrm{vec}(\mathbf{Y}) = \mathrm{vec}(\mathbf{R}^\mathrm{T}\mathbf{Y}\mathbf{C}). \tag{4}$$

We also exploit the compatibility of a Kronecker product plus a constant diagonal term with the eigenvalue decomposition:

$$(\mathbf{C} \otimes \mathbf{R} + \sigma^2\mathbf{I}) = (\mathbf{U_C} \otimes \mathbf{U_R})(\mathbf{S_C} \otimes \mathbf{S_R} + \sigma^2\mathbf{I})(\mathbf{U_C}^\mathrm{T} \otimes \mathbf{U_R}^\mathrm{T}), \tag{5}$$

where $\mathbf{C} = \mathbf{U_C}\mathbf{S_C}\mathbf{U_C}^\mathrm{T}$ is the eigenvalue decomposition of $\mathbf{C}$, and similarly for $\mathbf{R}$.

**Likelihood evaluation**   Using these identities, the log of the likelihood in Equation (3) follows as

$$\mathcal{L} = -\frac{N \cdot D}{2}\ln(2\pi) - \frac{1}{2}\ln\left|\mathbf{S_C} \otimes \mathbf{S_R} + \sigma^2\mathbf{I}\right|$$
$$- \frac{1}{2}\mathrm{vec}(\mathbf{U_R}^\mathrm{T}\mathbf{Y}\mathbf{U_C})^\mathrm{T}(\mathbf{S_C} \otimes \mathbf{S_R} + \sigma^2\mathbf{I})^{-1}\mathrm{vec}(\mathbf{U_R}^\mathrm{T}\mathbf{Y}\mathbf{U_C}). \tag{6}$$

This term can be interpreted as a multivariate normal distribution with diagonal covariance matrix $(\mathbf{S_C} \otimes \mathbf{S_R} + \sigma^2\mathbf{I})$ on rotated data $\mathrm{vec}(\mathbf{U_R}^\mathrm{T}\mathbf{Y}\mathbf{U_C})^\mathrm{T}$, similar to an approach that is used to speed up mixed models in genetics [13].

**Gradient evaluation**   Derivatives of the log marginal likelihood with respect to a particular covariance parameter $\theta_\mathbf{R} \in \mathbf{\Theta_R}$ can be expressed as

$$\frac{\mathrm{d}}{\mathrm{d}\theta_\mathbf{R}}\mathcal{L} = -\frac{1}{2}\mathrm{diag}\left((\mathbf{S_C} \otimes \mathbf{S_R} + \sigma^2\mathbf{I})^{-1}\right)^\mathrm{T}\mathrm{diag}\left(\mathbf{S_C} \otimes (\mathbf{U_R}^\mathrm{T}\frac{\mathrm{d}}{\mathrm{d}\theta_\mathbf{R}}\mathbf{R}\mathbf{U_R})\right)$$
$$+ \frac{1}{2}\mathrm{vec}(\tilde{\mathbf{Y}})^\mathrm{T}\mathrm{vec}\left(\mathbf{U_R}^\mathrm{T}\frac{\mathrm{d}}{\mathrm{d}\theta_\mathbf{R}}\mathbf{R}\mathbf{U_R}\tilde{\mathbf{Y}}\mathbf{S_C}\right), \tag{7}$$

where $\mathrm{vec}(\tilde{\mathbf{Y}}) = (\mathbf{S_C} \otimes \mathbf{S_R} + \sigma^2\mathbf{I})^{-1}\mathrm{vec}(\mathbf{U_R}^\mathrm{T}\mathbf{Y}\mathbf{U_C})$. Analogous expressions follow for partial derivatives with respect to $\theta_\mathbf{C} \in \mathbf{\Theta_C}$ and the noise level $\sigma^2$. Full details of all derivations, including derivatives wrt. $\sigma^2$, can be found in the supplementary material.

**Runtime and memory complexity**   A naïve implementation for optimizing the likelihood (3) with respect to the hyperparameters would have runtime complexity $\mathcal{O}(N^3D^3)$ and memory complexity $\mathcal{O}(N^2D^2)$. Using the likelihood and derivative as expressed in Equations (6) and (7), each evaluation with new kernel parameters involves solving the symmetric eigenvalue problems of both $\mathbf{R}$ and $\mathbf{C}$, together having a runtime complexity of $\mathcal{O}(N^3 + D^3)$. Explicit evaluation of any matrix Kronecker products is not necessary, resulting in a low memory complexity of $\mathcal{O}(N^2 + D^2)$.

# 3 Graphical Lasso in the presence of confounders

Estimation of sparse inverse covariance matrices is widely used to identify undirected network structures from observational data. However, non-iid observations due to hidden confounding variables may hinder accurate recovery of the true network structure. If not accounted for, confounders may lead to a large number of false positive edges. This is of particular relevance in biological applications, where observational data are often heterogeneous, combining measurements from different labs, data obtained under various perturbations or from a range of measurement platforms.

As an application of the framework described in Section 2, we here propose an approach to learning sparse inverse covariance matrices between features, while accounting for covariation between samples due to confounders. First, we briefly review the "orthogonal" approaches that account for the corresponding types of sample and feature covariance we set out to model.

## 3.1 Explaining feature dependencies using the Graphical Lasso

A common approach to model relationships between variables in a graphical model is the GLASSO. It has been used in the context of biological studies to recover the hidden network structure of gene-gene interrelationships [14], for instance. The GLASSO assumes a multivariate Gaussian distribution on features with a sparse precision (inverse covariance) matrix. The sparsity is induced by an $L_1$ penalty on the entries of $\mathbf{C}^{-1}$, the inverse of the feature covariance matrix.

Under the simplifying assumption of iid samples, the posterior distribution of $\mathbf{Y}$ under this model is proportional to

$$p(\mathbf{Y}, \mathbf{C}^{-1}) = p(\mathbf{C}^{-1}) \prod_{r=1}^{N} \mathcal{N}\left(\mathbf{Y}_{r,:} \,|\, \mathbf{0}_D, \mathbf{C}\right). \tag{8}$$

Here, the prior on the precision matrix $\mathbf{C}^{-1}$ is

$$p(\mathbf{C}^{-1}) \propto \exp\left(-\lambda \left\|\mathbf{C}^{-1}\right\|_1\right) [\mathbf{C}^{-1} \succ \mathbf{0}], \tag{9}$$

with $\|\mathbf{A}\|_1$ defined as the sum over all absolute values of the matrix entries. Note that this prior is only nonzero for positive-definite $\mathbf{C}^{-1}$.

## 3.2 Modeling confounders using the Gaussian process latent variable model

Confounders are unobserved variables that can lead to spurious associations between observed variables and to covariation between samples. A possible approach to identify such confounders is dimensionality reduction. Here we briefly review two dimensionality reduction methods, dual probabilistic PCA and its generalization, the Gaussian process latent variable model (GPLVM) [15]. In the context of applications, these methods have previously been applied to identify regulatory processes [16], and to recover confounding factors with broad effects on many features [11, 12].

In dual probabilistic PCA [15], the observed data $\mathbf{Y}$ is explained as a linear combination of $K$ latent variables ("factors"), plus independent observation noise. The model is as follows:

$$\mathbf{Y} = \mathbf{XW} + \mathbf{E},$$

where $\mathbf{X} \in \mathbb{R}^{N \times K}$ contains the values of $K$ latent variables ("factors"), $\mathbf{W} \in \mathbb{R}^{K \times D}$ contains independent standard-normally distributed weights that specify the mapping between latent and observed variables. Finally, $\mathbf{E} \in \mathbb{R}^{N \times D}$ contains iid Gaussian noise with $E_{rc} \sim \mathcal{N}(0, \sigma^2)$. Marginalizing over the weights $\mathbf{W}$ yields the data likelihood:

$$p(\mathbf{Y} \,|\, \mathbf{X}) = \prod_{c=1}^{D} \mathcal{N}\left(\mathbf{Y}_{:,c} \,|\, \mathbf{0}_N, \mathbf{XX}^{\mathrm{T}} + \sigma^2 \mathbf{I}_N\right). \tag{10}$$

Learning the latent factors $\mathbf{X}$ and the observation noise variance $\sigma^2$ can be done by maximum likelihood. The more general GPLVM [15] is obtained by replacing $\mathbf{XX}^{\mathrm{T}}$ in (10) with a more general Gram matrix $\mathbf{R}$, with $R_{rs} = \kappa\big((x_{r1}, \ldots, x_{rK}), (x_{s1}, \ldots, x_{sK})\big)$ for some covariance function $\kappa : \mathbb{R}^K \times \mathbb{R}^K \to \mathbb{R}$.

### 3.3 Combining the two models

We propose to combine these two different explanations of the data into one coherent model. Instead of treating either the samples or the features as being (conditionally) independent, we aim to learn a joint covariance for the observed data matrix $\mathbf{Y}$. This model, called Kronecker GLASSO, is a special instance of the Kronecker Gaussian process model introduced in Section 2, as the data likelihood can be written as:

$$p(\mathbf{Y} \,|\, \mathbf{R}, \mathbf{C}^{-1}) = \mathcal{N}\left(\mathrm{vec}(\mathbf{Y}) \,\big|\, \mathbf{0}_{N \cdot D}, \mathbf{C} \otimes \mathbf{R} + \sigma^2 \mathbf{I}_{N \cdot D}\right). \tag{11}$$

Here, we build on the model components introduced in Section 3.2 and Section 3.1. We use the sparse $L_1$ penalty (9) for the feature inverse covariance $\mathbf{C}^{-1}$ and use a linear kernel for the covariance on rows $\mathbf{R} = \mathbf{X}\mathbf{X}^{\mathrm{T}} + \rho^2 \mathbf{I}_N$. Learning the model parameters proceeds via MAP inference, optimizing the log likelihood implied by Equation (11) with respect to $\mathbf{X}$ and $\mathbf{C}^{-1}$, and the hyperparameters $\sigma^2$, $\rho^2$. By combining the GLASSO and GPLVM in this way, we can recover network structure in the presence of confounders.

An equivalent generative model can be obtained in a similar way as in dual probabilistic PCA. The main difference is that now, the rows of the weight matrix $\mathbf{W}$ are sampled from a $\mathcal{N}(\mathbf{0}_D, \mathbf{C})$ distribution instead of a $\mathcal{N}(\mathbf{0}_D, \mathbf{I}_D)$ distribution. This generative model for $\mathbf{Y}$ given latent variables $\mathbf{X} \in \mathbb{R}^{N \times K}$ and feature covariance $\mathbf{C} \in \mathbb{R}^{D \times D}$ is of the form $\mathbf{Y} = \mathbf{X}\mathbf{W} + \rho\mathbf{V} + \mathbf{E}$, where $\mathbf{W} \in \mathbb{R}^{K \times D}$, $\mathbf{V} \in \mathbb{R}^{N \times D}$ and $\mathbf{E} \in \mathbb{R}^{N \times D}$ are jointly independent with distributions $\mathrm{vec}(\mathbf{W}) \sim \mathcal{N}(\mathbf{0}_{KD}, \mathbf{C} \otimes \mathbf{I}_K)$, $\mathrm{vec}(\mathbf{V}) \sim \mathcal{N}(\mathbf{0}_{ND}, \mathbf{C} \otimes \mathbf{I}_N)$ and $\mathrm{vec}(\mathbf{E}) \sim \mathcal{N}(\mathbf{0}_{ND}, \sigma^2 \mathbf{I}_{ND})$.

### 3.4 Inference in the joint model

As already mentioned in Section 2, parameter inference in the Kronecker GLASSO model implied by Equation (11), when done naïvely, is intractable for all but very low dimensional data matrices $\mathbf{Y}$. Even using the tricks discussed in Section 2, free-form sparse inverse covariance updates for $\mathbf{C}^{-1}$ are intractable under the $L_1$ penalty when depending on gradient updates.

Similar as in Section 2, the first step towards efficient inference is to introduce $N \times D$ additional latent variables $\mathbf{Z}$, which can be thought of as the noise-free observations:

$$p(\mathbf{Y}|\mathbf{Z}, \sigma^2) = \mathcal{N}\left(\mathrm{vec}(\mathbf{Y}) \,\big|\, \mathrm{vec}(\mathbf{Z}), \sigma^2 \mathbf{I}_{N \cdot D}\right) \tag{12}$$

$$p(\mathbf{Z}|\mathbf{R}, \mathbf{C}) = \mathcal{N}\left(\mathrm{vec}(\mathbf{Z}) \,|\, \mathbf{0}_{N \cdot D}, \mathbf{C} \otimes \mathbf{R}\right). \tag{13}$$

We consider the latent variables $\mathbf{Z}$ as additional model parameters. We now optimize the distribution $p(\mathbf{Y}, \mathbf{C}^{-1} \,|\, \mathbf{Z}, \mathbf{R}, \sigma^2) = p(\mathbf{Y} \,|\, \mathbf{Z}, \sigma^2)p(\mathbf{Z} \,|\, \mathbf{R}, \mathbf{C})p(\mathbf{C}^{-1})$ with respect to the unknown parameters $\mathbf{Z}$, $\mathbf{C}^{-1}$, $\sigma^2$, and $\mathbf{R}$ (which depends on $\mathbf{X}$ and kernel parameters $\mathbf{\Theta}_{\mathbf{R}}$) by iterating through the following steps:

1. Optimize for $\sigma^2, \mathbf{R}$ after integrating out $\mathbf{Z}$, for fixed $\mathbf{C}$:

   $$\underset{\sigma^2, \mathbf{\Theta}_{\mathbf{R}}, \mathbf{X}}{\mathrm{argmax}} \, p(\mathbf{Y} \,|\, \mathbf{C}, \mathbf{R}(\mathbf{\Theta}_{\mathbf{R}}, \mathbf{X}), \sigma^2) =$$
   $$\underset{\sigma^2, \mathbf{\Theta}_{\mathbf{R}}, \mathbf{X}}{\mathrm{argmax}} \, \mathcal{N}\left(\mathrm{vec}(\mathbf{Y}) \,\big|\, \mathbf{0}_{N \cdot D}, \mathbf{C} \otimes \mathbf{R}(\mathbf{\Theta}_{\mathbf{R}}, \mathbf{X}) + \sigma^2 \mathbf{I}_{N \cdot D}\right) \tag{14}$$

2. Calculate the expectation of $\mathbf{Z}$ for fixed $\mathbf{R}$, $\mathbf{C}$, and $\sigma^2$ :

   $$\mathrm{vec}(\hat{\mathbf{Z}}) = (\mathbf{C} \otimes \mathbf{R})(\mathbf{C} \otimes \mathbf{R} + \sigma^2 \mathbf{I}_{N \cdot D})^{-1} \mathrm{vec}(\mathbf{Y})$$

3. Optimize $\hat{\mathbf{C}}^{-1}$ for fixed $\mathbf{R}$ and $\hat{\mathbf{Z}}$:

   $$\underset{\hat{\mathbf{C}}^{-1}}{\mathrm{argmax}} \, p(\hat{\mathbf{C}}^{-1} \,|\, \hat{\mathbf{Z}}, \mathbf{R}) = \underset{\hat{\mathbf{C}}^{-1}}{\mathrm{argmax}} \, \mathcal{N}\left(\mathrm{vec}(\hat{\mathbf{Z}}) \,\big|\, \mathbf{0}, \hat{\mathbf{C}} \otimes \mathbf{R}\right) p(\hat{\mathbf{C}}^{-1})$$

   and set $\mathbf{C} = \hat{\mathbf{C}}$.

As a stopping criterion we consider the relative reduction of the negative log-marginal likelihood (Equation (11)) plus the regularizer on $\mathbf{C}^{-1}$. The choice to optimize $\hat{\mathbf{C}}^{-1}$ for fixed $\hat{\mathbf{Z}}$ is motivated by computational considerations, as this subproblem then reduces to conventional GLASSO; a full EM approach with latent variables $\mathbf{Z}$ does not seem feasible. Step 1 can be done using the efficient likelihood evaluations and gradients described in Section 2. We will now discuss step 3 in more detail.

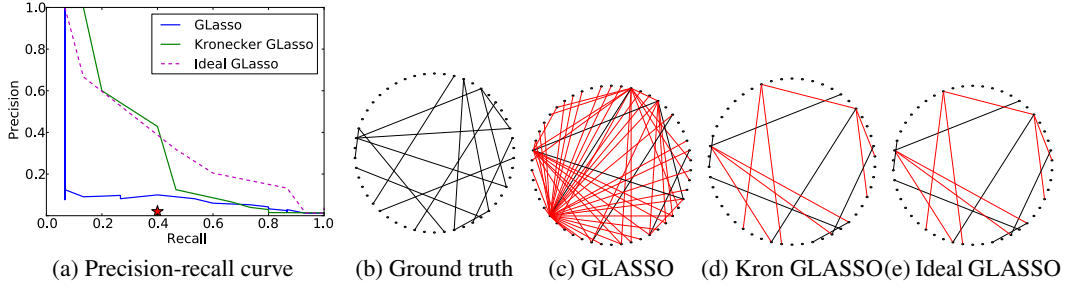

(a) Precision-recall curve     (b) Ground truth    (c) GLASSO    (d) Kron GLASSO(e) Ideal GLASSO

Figure 1: Network reconstruction on the simulated example. **(a)** Precision-recall curve, when varying the sparsity penalty $\lambda$. Compared are the standard GLASSO, our algorithm with Kronecker structure (Kronecker GLASSO) and as a reference an idealized setting, applying standard GLASSO to a similar dataset without confounding influences (Ideal GLASSO). The model that accounts for confounders approaches the performance of an idealized model, while standard GLASSO finds a large fraction of false positive edges. **(b)** Ground truth network. **(c-e)** Recovered networks for GLASSO, Kronecker GLASSO and Ideal GLASSO at 40% recall (star in **(a)**). False positive predicted edges are colored in red. Because of the effect of confounders, standard GLASSO predicted an excess of edges to 4 of the nodes.

**Optimizing for $\hat{\mathbf{C}}^{-1}$**    The third step, optimizing with respect to $\hat{\mathbf{C}}^{-1}$, can be done efficiently, using similar ideas as in Section 2. First consider:

$$\ln \mathcal{N}\left(\mathrm{vec}(\hat{\mathbf{Z}}) \,\middle|\, \mathbf{0}_{N \cdot D}, \hat{\mathbf{C}} \otimes \mathbf{R}\right) = -\frac{N \cdot D}{2}\ln(2\pi) - \frac{1}{2}\ln\left|\hat{\mathbf{C}} \otimes \mathbf{R}\right| - \frac{1}{2}\mathrm{vec}(\hat{\mathbf{Z}})^{\mathrm{T}}(\hat{\mathbf{C}} \otimes \mathbf{R})^{-1}\mathrm{vec}(\hat{\mathbf{Z}}).$$

Now, using the Kronecker identity (4) and

$$\ln|\mathbf{A} \otimes \mathbf{B}| \quad = \mathrm{rank}(\mathbf{B})\ln|\mathbf{A}| + \mathrm{rank}(\mathbf{A})\ln|\mathbf{B}|,$$

we can rewrite the log likelihood as:

$$\ln \mathcal{N}\left(\mathrm{vec}(\hat{\mathbf{Z}}) \,\middle|\, \mathbf{0}, \hat{\mathbf{C}} \otimes \mathbf{R}\right) p(\hat{\mathbf{C}}^{-1})$$

$$= -\frac{N \cdot D}{2}\ln(2\pi) - \frac{1}{2}D\ln|\mathbf{R}| + \frac{1}{2}N\ln\left|\hat{\mathbf{C}}^{-1}\right| - \frac{1}{2}\mathrm{Tr}(\hat{\mathbf{Z}}^{\mathrm{T}}\mathbf{R}^{-1}\hat{\mathbf{Z}}\hat{\mathbf{C}}^{-1}).$$

Thus we obtain a standard GLASSO problem with covariance matrix $\hat{\mathbf{Z}}^{\mathrm{T}}\mathbf{R}^{-1}\hat{\mathbf{Z}}$:

$$\underset{\hat{\mathbf{C}}^{-1}}{\mathrm{argmax}}\, p(\hat{\mathbf{C}}^{-1} \,|\, \hat{\mathbf{Z}}, \mathbf{R}) = \underset{\hat{\mathbf{C}}^{-1} \succ 0}{\mathrm{argmax}}\left(-\frac{1}{2}\mathrm{Tr}(\hat{\mathbf{Z}}^{\mathrm{T}}\mathbf{R}^{-1}\hat{\mathbf{Z}}\hat{\mathbf{C}}^{-1}) + \frac{1}{2}N\ln\left|\hat{\mathbf{C}}^{-1}\right| - \lambda\left\|\hat{\mathbf{C}}^{-1}\right\|_1\right). \quad (15)$$

The inverse sample covariance $\mathbf{R}^{-1}$ in Equation (15) rotates the data covariance, similar as in the established flip-flop algorithm for inference in matrix-variate normal distributions [7, 1].

## 4   Experiments

In this Section, we describe three experiments with the generalized GLASSO.

### 4.1   Simulation study

First, we considered an artificial dataset to illustrate the effect of confounding factors on the solution quality of sparse inverse covariance estimation. We created synthetic data, with $N = 100$ samples and $D = 50$ features according to the generative model described in Section 3.3. We generated the sparse inverse column covariance $\mathbf{C}^{-1}$ choosing edges at random with a sparsity level of $1\%$. Non-zero entries of the inverse covariance were drawn from a Gaussian with mean 1 and variance 2. The row covariance matrix $\mathbf{R}$ was created from $K = 3$ random factors $\mathbf{x}_k$, each drawn from unit variance iid Gaussian variables. The weighting between the confounders and the iid component $\rho^2$ was set such that the factors explained equal variance, which corresponds to moderate extent of confounding influences. Finally, we added independent Gaussian observation noise, choosing a signal-to-noise ratio of $10\%$.

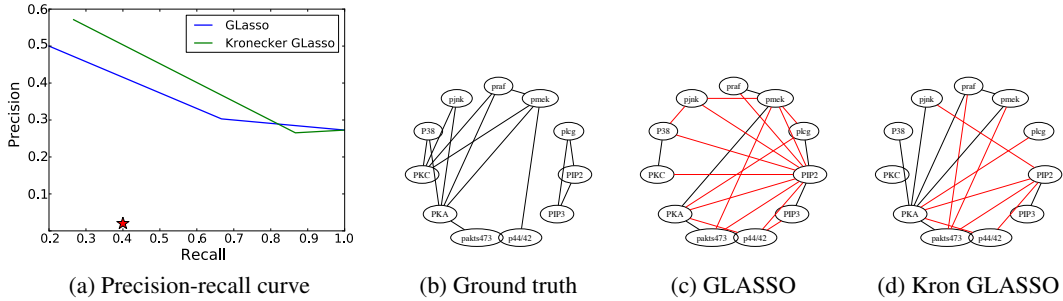

|     |     |     |     |
| --- | --- | --- | --- |
| (a) Precision-recall curve | (b) Ground truth | (c) GLASSO | (d) Kron GLASSO |

Figure 2: Network reconstruction of a protein signaling network from Sachs et al. **(a)** Precision-recall curve, when varying the sparsity penalty $\lambda$. Compared are the standard GLASSO, and our algorithm with Kronecker structure (Kronecker GLASSO). Standard GLASSO, not accounting for confounders, found more false positive edges for a wide range of recall rates. **(b)** Ground truth network. **(c-d)** Recovered networks for GLASSO and Kronecker GLASSO at 40% recall (star in **(a)**). False positive edge predictions are colored in red.

Next, we applied different methods to reconstruct the true simulated network. We considered standard GLASSO and our Kronecker model that accounts for the confounding influence (Kronecker GLASSO). For reference, we also considered an idealized setting, applying GLASSO to a similar dataset without the confounding effects (Ideal GLASSO), obtained by setting $\mathbf{X} = \mathbf{0}_{N \cdot K}$ in the generative model. To determine an appropriate latent dimensionality of Kronecker GLASSO, we used the BIC criterion on multiple restarts with $K = 1$ to $K = 5$ latent factors. For all models we varied the sparsity parameter of the graphical lasso, setting $\lambda = 5^x$, with $x$ linearly interpolated between $-8$ and $3$. The solution set of lasso-based algorithms is typically unstable and depends on slight variation of the data. To improve the stability of all methods, we employed stability selection [17], applying each algorithm for all regularization parameters 100 times to randomly drawn subsets containing 90% of the data. We then considered edges that were found in at least 50% of all 100 restarts.

Figure 1a shows the precision-recall curve for each algorithm. Kronecker GLASSO performed considerably better than standard GLASSO, approaching the performance of the ideal model without confounders. Figures 1b-d show the reconstructed networks at 40% recall. While Kronecker GLASSO reconstructed the same network as the ideal model, standard GLASSO found an excess of false positive edges.

## 4.2 Network reconstruction of protein-signaling networks

Important practical applications of the GLASSO include the reconstruction of gene and protein networks. Here, we revisit the extensively studied protein signaling data from Sachs et al. [18]. The dataset provides observational data of the activations of 11 proteins under various external stimuli. We combined measurements from the first 3 experiments, yielding a heterogeneous mix of 2,666 samples that are not expected to be an iid sample set. To make the inference more difficult, we selected a random fraction of 10% of the samples, yielding a final data matrix of size 266 times 11. We used the directed ground truth network and moralized the graph structure to obtain an undirected ground truth network. Parameter choice and stability selection were done as in the simulation study.

Figure 2 shows the results. Analogous to the simulation setting, the Kronecker GLASSO model found true network links with greater accuracy than standard graphical lasso. This results suggest that our model is suitable to account for confounding variation as it occurs in real settings.

## 4.3 Large-scale application to yeast gene expression data

Next, we considered an application to large-scale gene expression profiling data from yeast. We revisited the dataset from Smith et al. [19], consisting of 109 genetically diverse yeast strains, each of which has been expression profiled in two environmental conditions (glucose and ethanol). Because

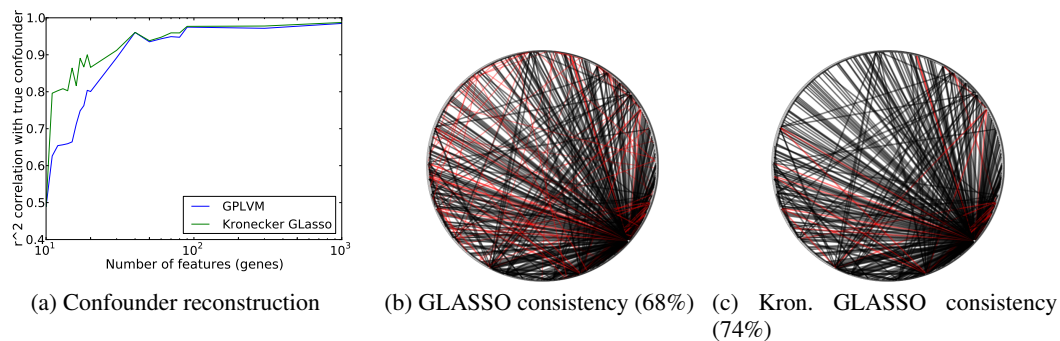

| (a) Confounder reconstruction | (b) GLASSO consistency (68%) | (c) Kron. GLASSO consistency (74%) |

Figure 3: **(a)** Correlation coefficient between learned confounding factor and true environmental condition for different subsets of all features (genes). Compared are the standard GPLVM model with a linear covariance and our proposed model that accounts for low rank confounders and sparse gene-gene relationships (Kronecker GLASSO). Kronecker GLASSO is able to better recover the hidden confounder by accounting for the covariance structure between genes. **(b,c)** Consistency of edges on the largest network with 1,000 nodes learnt on the joint dataset, comparing the results when combining both conditions with those for a single condition (glucose).

the confounder in this dataset is known explicitly, we tested the ability of Kronecker GLASSO to recover it from observational data. Because of missing complete ground truth information, we could not evaluate the network reconstruction quality directly. An appropriate regularization parameter was selected by means of cross validation, evaluating the marginal likelihood on a test set (analogous to the procedure described in [10]). To simplify the comparison to the known confounding factor, we chose a fixed number of confounders that we set to $K = 1$.

**Recovery of the known confounder**   Figure 3a shows the $r^2$ correlation coefficient between the inferred factor and the true environmental condition for increasing number of features (genes) that were used for learning. In particular for small numbers of genes, accounting for the network structure between genes improved the ability to recover the true confounding effect.

**Consistency of obtained networks**   Next, we tested the consistency when applying GLASSO and Kronecker GLASSO to data that combines both conditions, glucose and ethanol, comparing to the recovered network from a single condition alone (glucose). The respective networks are shown in Figures 3b and 3c. The Kronecker GLASSO model identifies more consistent edges, which shows the susceptibility of standard GLASSO to the confounder, here the environmental influence.

## 5   Conclusions and Discussion

We have shown an efficient scheme for parameter learning in matrix-variate normal distributions with iid observation noise. By exploiting some linear algebra tricks, we have shown how hyper-parameter optimization for the row and column covariances can be carried out without evaluating the prohibitive full covariance, thereby greatly reducing computational and memory complexity. To the best of our knowledge, these measures have not previously been proposed, despite their general applicability.

As an application of our framework, we have proposed a method that accounts for confounding influences while estimating a sparse inverse covariance structure. Our approach extends the Graphical Lasso, generalizing the rigid assumption of iid samples to more general sample covariances. For this purpose, we employ a Kronecker product covariance structure and learn a low-rank covariance between samples, thereby accounting for potential confounding influences. We provided synthetic and real world examples where our method is of practical use, reducing the number of false positive edges learned.

**Acknowledgments**   This research was supported by the FP7 PASCAL II Network of Excellence. OS received funding from the Volkswagen Foundation. JM was supported by NWO, the Netherlands Organization for Scientific Research (VENI grant 639.031.036).

## Footnotes

[1]These authors contributed equally to this work.

# References

[1] Y. Zhang and J. Schneider. Learning multiple tasks with a sparse matrix-normal penalty. In *Advances in Neural Information Processing Systems*, 2010.

[2] E. Bonilla, K.M. Chai, and C. Williams. Multi-task gaussian process prediction. *Advances in Neural Information Processing Systems*, 20:153–160, 2008.

[3] M.A. Alvarez and N.D. Lawrence. Computationally efficient convolved multiple output gaussian processes. *Journal of Machine Learning Research*, 12:1425–1466, 2011.

[4] H. Wackernagel. *Multivariate geostatistics: an introduction with applications*. Springer Verlag, 2003.

[5] G.I. Allen and R. Tibshirani. Inference with transposable data: Modeling the effects of row and column correlations. *Arxiv preprint arXiv:1004.0209*, 2010.

[6] M. Lynch and B. Walsh. *Genetics and Analysis of Quantitative Traits*. Sinauer Associates Inc., U.S., 1998.

[7] P. Dutilleul. The MLE algorithm for the matrix normal distribution. *Journal of Statistical Computation and Simulation*, 64(2):105–123, 1999.

[8] K. Zhang, B. Schölkopf, and D. Janzing. Invariant gaussian process latent variable models and application in causal discovery. In *Uncertainty in Artificial Intelligence*, 2010.

[9] O. Banerjee, L. El Ghaoui, and A. d'Aspremont. Model selection through sparse maximum likelihood estimation for multivariate gaussian or binary data. *Journal of Machine Learning Research*, 9:485–516, 2008.

[10] J. Friedman, T. Hastie, and R. Tibshirani. Sparse inverse covariance estimation with the graphical lasso. *Biostatistics*, 9(3):432, 2008.

[11] J.T. Leek and J.D. Storey. Capturing heterogeneity in gene expression studies by surrogate variable analysis. *PLoS Genetics*, 3(9):e161, 2007.

[12] O. Stegle, L. Parts, R. Durbin, and J. Winn. A bayesian framework to account for complex non-genetic factors in gene expression levels greatly increases power in eqtl studies. *PLoS Computational Biology*, 6(5):e1000770, 2010.

[13] C. Lippert, J. Listgarten, Y. Liu, C.M. Kadie, R.I. Davidson, and D. Heckerman. FaST linear mixed models for genome-wide association studies. *Nature Methods*, 8:833–835, 2011.

[14] P. Menéndez, Y.A.I. Kourmpetis, C.J.F. Ter Braak, and F.A. van Eeuwijk. Gene regulatory networks from multifactorial perturbations using graphical lasso: Application to the dream4 challenge. *PLoS One*, 5(12):e14147, 2010.

[15] N. Lawrence. Probabilistic non-linear principal component analysis with gaussian process latent variable models. *Journal of Machine Learning Research*, 6:1783–1816, 2005.

[16] K.Y. Yeung and W.L. Ruzzo. Principal component analysis for clustering gene expression data. *Bioinformatics*, 17(9):763, 2001.

[17] N. Meinshausen and P. Bühlmann. Stability selection. *Journal of the Royal Statistical Society: Series B (Statistical Methodology)*, 72(4):417–473, 2010.

[18] K. Sachs, O. Perez, D. Pe'er, D.A. Lauffenburger, and G.P. Nolan. Causal protein-signaling networks derived from multiparameter single-cell data. *Science*, 308(5721):523, 2005.

[19] E.N. Smith and L. Kruglyak. Gene–environment interaction in yeast gene expression. *PLoS Biology*, 6(4):e83, 2008.

